# A Unified Near-Optimal Estimator For Dimension Reduction in $l_\alpha$ ($0 < \alpha \le 2$) Using Stable Random Projections

**Ping Li**
Department of Statistical Science
Faculty of Computing and Information Science
Cornell University
pingli@cornell.edu

**Trevor J. Hastie**
Department of Statistics
Department of Health, Research and Policy
Stanford University
hastie@stanford.edu

## Abstract

Many tasks (e.g., clustering) in machine learning only require the $l_\alpha$ distances instead of the original data. For dimension reductions in the $l_\alpha$ norm ($0 < \alpha \le 2$), the method of *stable random projections* can efficiently compute the $l_\alpha$ distances in massive datasets (e.g., the Web or massive data streams) in one pass of the data. The estimation task for *stable random projections* has been an interesting topic. We propose a simple estimator based on the *fractional power* of the samples (projected data), which is surprisingly near-optimal in terms of the asymptotic variance. In fact, it achieves the Cramér-Rao bound when $\alpha = 2$ and $\alpha = 0+$. This new result will be useful when applying *stable random projections* to distance-based clustering, classifications, kernels, massive data streams etc.

## 1 Introduction

Dimension reductions in the $l_\alpha$ norm ($0 < \alpha \le 2$) have numerous applications in data mining, information retrieval, and machine learning. In modern applications, the data can be way too large for the physical memory or even the disk; and sometimes only one pass of the data can be afforded for building statistical learning models [1, 2, 5]. We abstract the data as a *data matrix* $\mathbf{A} \in \mathbb{R}^{n \times D}$. In many applications, it is often the case that we only need the $l_\alpha$ properties (norms or distances) of $\mathbf{A}$. The method of *stable random projections* [9, 18, 22] is a useful tool for efficiently computing the $l_\alpha$ ($0 < \alpha \le 2$) properties in massive data using a small (memory) space.

Denote the leading two rows in the data matrix $\mathbf{A}$ by $u_1$, $u_2 \in \mathbb{R}^D$. The $l_\alpha$ distance $d_{(\alpha)}$ is

$$d_{(\alpha)} = \sum_{i=1}^{D} |u_{1,i} - u_{2,i}|^\alpha. \tag{1}$$

The choice of $\alpha$ is beyond the scope of this study; but basically, we can treat $\alpha$ as a *tuning* parameter. In practice, the most popular choice, i.e., the $\alpha = 2$ norm, often does not work directly on the original (unweighted) data, as it is well-known that truly large-scale datasets (especially Internet data) are ubiquitously "heavy-tailed." In machine learning, it is often crucial to carefully *term-weight* the data (e.g., taking logarithm or tf-idf) before applying subsequent learning algorithms using the $l_2$ norm. As commented in [12, 21], the *term-weighting* procedure is often far more important than fine-tuning the learning parameters. Instead of weighting the original data, an alternative scheme is to choose an appropriate norm. For example, the $l_1$ norm has become popular recently, e.g., LASSO, LARS, 1-norm SVM [23], Laplacian radial basis kernel [4], etc. But other norms are also possible. For example, [4] proposed a family of non-Gaussian radial basis kernels for SVM in the form $K(x, y) = \exp\left(-\rho \sum_i |x_i - y_i|^\alpha\right)$, where $x$ and $y$ are data points in high-dimensions; and [4] showed that $\alpha \le 1$ (even $\alpha = 0$) in some cases produced better results in histogram-based image classifications. The $l_\alpha$ norm with $\alpha < 1$, which may initially appear strange, is now well-understood to be a natural measure of sparsity [6]. In the extreme case, when $\alpha \to 0+$, the $l_\alpha$ norm approaches the Hamming norm (i.e., the number of non-zeros in the vector).

Therefore, there is the natural demand in science and engineering for dimension reductions in the $l_\alpha$ norm other than $l_2$. While the method of *normal random projections* for the $l_2$ norm [22] has become very popular recently, we have to resort to more general methodologies for $0 < \alpha < 2$. The idea of *stable random projections* is to multiply $\mathbf{A}$ with a random projection matrix $\mathbf{R} \in \mathbb{R}^{D \times k}$ ($k \ll D$). The matrix $\mathbf{B} = \mathbf{A} \times \mathbf{R} \in \mathbb{R}^{n \times k}$ will be much smaller than $\mathbf{A}$. The entries of $\mathbf{R}$ are (typically) i.i.d. samples from a symmetric $\alpha$-stable distribution [24], denoted by $S(\alpha, 1)$, where $\alpha$ is the index and 1 is the scale. We can then discard the original data matrix $\mathbf{A}$ because the projected matrix $\mathbf{B}$ now contains enough information to recover the original $l_\alpha$ properties approximately.

A symmetric $\alpha$-stable random variable is denoted by $S(\alpha, d)$, where $d$ is the scale parameter. If $x \sim S(\alpha, d)$, then its characteristic function (Fourier transform of the density function) would be

$$\mathrm{E}\left(\exp\left(\sqrt{-1}xt\right)\right) = \exp\left(-d|t|^{\alpha}\right), \tag{2}$$

whose inverse does not have a closed-form except for $\alpha = 2$ (i.e., normal) or $\alpha = 1$ (i.e., Cauchy).

Applying stable random projections on $u_1 \in \mathbb{R}^D$, $u_2 \in \mathbb{R}^D$ yields, respectively, $v_1 = \mathbf{R}^{\mathbf{T}}u_1 \in \mathbb{R}^k$ and $v_2 = \mathbf{R}^{\mathbf{T}}u_2 \in \mathbb{R}^k$. By the properties of Fourier transforms, the projected differences, $v_{1,j} - v_{2,j}$, $j = 1, 2, ..., k$, are i.i.d. samples of the stable distribution $S(\alpha, d_{(\alpha)})$, i.e.,

$$x_j = v_{1,j} - v_{2,j} \sim S(\alpha, d_{(\alpha)}), \quad j = 1, 2, ..., k. \tag{3}$$

Thus, the task is to estimate the scale parameter from $k$ i.i.d. samples $x_j \sim S(\alpha, d_{(\alpha)})$. Because no closed-form density functions are available except for $\alpha = 1, 2$, the estimation task is challenging when we seek estimators that are both accurate and computationally efficient.

For general $0 < \alpha < 2$, a widely used estimator is based on the sample *inter-quantiles* [7,20], which can be simplified to be the *sample median* estimator by choosing the $0.75$ - $0.25$ sample quantiles and using the symmetry of $S(\alpha, d_{(\alpha)})$. That is

$$\hat{d}_{(\alpha),me} = \frac{\mathrm{median}\{|x_j|^{\alpha}, j = 1, 2, ..., k\}}{\mathrm{median}\{S(\alpha, 1)\}^{\alpha}}. \tag{4}$$

It has been well-known that the *sample median* estimator is not accurate, especially when the sample size $k$ is not too large. Recently, [13] proposed various estimators based on the geometric mean and the harmonic mean of the samples. The *harmonic mean* estimator only works for small $\alpha$. The *geometric mean* estimator has nice properties including closed-form variances, closed-form tail bounds in exponential forms, and very importantly, an analog of the Johnson-Lindenstrauss (JL) Lemma [10] for dimension reduction in $l_{\alpha}$. The *geometric mean estimator*, however, can still be improved for certain $\alpha$, especially for large samples (e.g., as $k \to \infty$).

## 1.1   Our Contribution: the *Fractional Power* Estimator

The *fractional power* estimator, with a simple unified format for all $0 < \alpha \le 2$, is (surprisingly) near-optimal in the Cramér-Rao sense (i.e., when $k \to \infty$, its variance is close to the Cramér-Rao lower bound). In particularly, it achieves the Cramér-Rao bound when $\alpha = 2$ and $\alpha \to 0+$.

The basic idea is straightforward. We first obtain an unbiased estimator of $d_{(\alpha)}^{\lambda}$, denoted by $\hat{R}_{(\alpha),\lambda}$. We then estimate $d_{(\alpha)}$ by $\left(\hat{R}_{(\alpha),\lambda}\right)^{1/\lambda}$, which can be improved by removing the $O\left(\frac{1}{k}\right)$ bias (this consequently also reduces the variance) using Taylor expansions. We choose $\lambda = \lambda^*(\alpha)$ to minimize the theoretical asymptotic variance. We prove that $\lambda^*(\alpha)$ is the solution to a simple convex program, i.e., $\lambda^*(\alpha)$ can be pre-computed and treated as a constant for every $\alpha$. The main computation involves only $\left(\sum_{j=1}^{k}|x_j|^{\lambda^*\alpha}\right)^{1/\lambda^*}$; and hence this estimator is also computationally efficient.

## 1.2   Applications

The method of *stable random projections* is useful for efficiently computing the $l_{\alpha}$ properties (norms or distances) in massive data, using a small (memory) space.

- *Data stream computations*    Massive data streams are fundamental in many modern data processing application [1, 2, 5, 9]. It is common practice to store only a very small *sketch* of the streams to efficiently compute the $l_{\alpha}$ norms of the individual streams or the $l_{\alpha}$ distances between a pair of streams. For example, in some cases, we only need to visually monitor the time history of the $l_{\alpha}$ distances; and approximate answers often suffice.

  One interesting special case is to estimate the Hamming norms (or distances) using the fact that, when $\alpha \to 0+$, $d_{(\alpha)} = \sum_{i=1}^{D}|u_{1,i} - u_{2,i}|^{\alpha}$ approaches the total number of non-zeros in $\{|u_{1,i} - u_{2,i}|\}_{i=1}^{D}$, i.e., the Hamming distance [5]. One may ask why not just (binary) quantize the data and then apply *normal random projections* to the binary data. [5] considered that the data are *dynamic* (i.e., frequent addition/subtraction) and hence pre-quantizing the data would not work. With *stable random projections*, we only need to update the corresponding sketches whenever the data are updated.

- *Computing all pairwise distances*    In many applications including distanced-based clustering, classifications and kernels (e.g.) for SVM, we only need the pairwise distances. Computing all pairwise distances of $\mathbf{A} \in \mathbb{R}^{n \times D}$ would cost $O(n^2 D)$, which can be significantly reduced to $O(nDk + n^2 k)$ by *stable random projections*. The cost reduction will be more considerable when the original datasets are too large for the physical memory.

- *Estimating $l_\alpha$ distances online*    While it is often infeasible to store the original matrix $\mathbf{A}$ in the memory, it is also often infeasible to materialize all pairwise distances in $\mathbf{A}$. Thus, in applications such as online learning, databases, search engines, online recommendation systems, and online market-basket analysis, it is often more efficient if we store $\mathbf{B} \in \mathbb{R}^{n \times k}$ in the memory and estimate any pairwise distance in $\mathbf{A}$ *on the fly* only when it is necessary.

When we treat $\alpha$ as a tuning parameter, i.e., re-computing the $l_\alpha$ distances for many different $\alpha$, *stable random projections* will be even more desirable as a cost-saving device.

## 2  Previous Estimators

We assume $k$ i.i.d. samples $x_j \sim S(\alpha, d_{(\alpha)})$, $j = 1, 2, ..., k$. We list several previous estimators.

- The *geometric mean* estimator is recommended in [13] for $\alpha < 2$.

$$\hat{d}_{(\alpha),gm} = \frac{\prod_{j=1}^{k} |x_j|^{\alpha/k}}{\left[\frac{2}{\pi}\Gamma\left(\frac{\alpha}{k}\right)\Gamma\left(1-\frac{1}{k}\right)\sin\left(\frac{\pi}{2}\frac{\alpha}{k}\right)\right]^k}. \tag{5}$$

$$\text{Var}\left(\hat{d}_{(\alpha),gm}\right) = d_{(\alpha)}^2 \left\{ \frac{\left[\frac{2}{\pi}\Gamma\left(\frac{2\alpha}{k}\right)\Gamma\left(1-\frac{2}{k}\right)\sin\left(\pi\frac{\alpha}{k}\right)\right]^k}{\left[\frac{2}{\pi}\Gamma\left(\frac{\alpha}{k}\right)\Gamma\left(1-\frac{1}{k}\right)\sin\left(\frac{\pi}{2}\frac{\alpha}{k}\right)\right]^{2k}} - 1 \right\} \tag{6}$$

$$= d_{(\alpha)}^2 \left\{ \frac{1}{k}\frac{\pi^2}{12}\left(\alpha^2 + 2\right) \right\} + O\left(\frac{1}{k^2}\right). \tag{7}$$

- The *harmonic mean* estimator is recommended in [13] for $0 < \alpha \leq 0.344$.

$$\hat{d}_{(\alpha),hm} = \frac{-\frac{2}{\pi}\Gamma(-\alpha)\sin\left(\frac{\pi}{2}\alpha\right)}{\sum_{j=1}^{k}|x_j|^{-\alpha}}\left(k - \left(\frac{-\pi\Gamma(-2\alpha)\sin(\pi\alpha)}{\left[\Gamma(-\alpha)\sin\left(\frac{\pi}{2}\alpha\right)\right]^2} - 1\right)\right), \tag{8}$$

$$\text{Var}\left(\hat{d}_{(\alpha),hm}\right) = d_{(\alpha)}^2 \frac{1}{k}\left(\frac{-\pi\Gamma(-2\alpha)\sin(\pi\alpha)}{\left[\Gamma(-\alpha)\sin\left(\frac{\pi}{2}\alpha\right)\right]^2} - 1\right) + O\left(\frac{1}{k^2}\right). \tag{9}$$

- For $\alpha = 2$, the *arithmetic mean* estimator, $\frac{1}{k}\sum_{j=1}^{k}|x_j|^2$, is commonly used, which has variance $= \frac{2}{k}d_{(2)}^2$. It can be improved by taking advantage of the marginal $l_2$ norms [17].

## 3  The Fractional Power Estimator

The *fractional power* estimator takes advantage of the following statistical result in Lemma 1.

**Lemma 1**  *Suppose $x \sim S\left(\alpha, d_{(\alpha)}\right)$. Then for $-1 < \lambda < \alpha$,*

$$E\left(|x|^\lambda\right) = d_{(\alpha)}^{\lambda/\alpha}\frac{2}{\pi}\Gamma\left(1-\frac{\lambda}{\alpha}\right)\Gamma(\lambda)\sin\left(\frac{\pi}{2}\lambda\right). \tag{10}$$

*If $\alpha = 2$, i.e., $x \sim S(2, d_{(2)}) = N(0, 2d_{(2)})$, then for $\lambda > -1$,*

$$E\left(|x|^\lambda\right) = d_{(2)}^{\lambda/2}\frac{2}{\pi}\Gamma\left(1-\frac{\lambda}{2}\right)\Gamma(\lambda)\sin\left(\frac{\pi}{2}\lambda\right) = d_{(2)}^{\lambda/2}\frac{2\Gamma(\lambda)}{\Gamma\left(\frac{\lambda}{2}\right)}. \tag{11}$$

**Proof:**  *For $0 < \alpha \leq 2$ and $-1 < \lambda < \alpha$, (10) can be inferred directly from [24, Theorem 2.6.3]. For $\alpha = 2$, the moment $E\left(|x|^\lambda\right)$ exists for any $\lambda > -1$. (11) can be shown by directly integrating the Gaussian density (using the integral formula [8, 3.381.4]). The Euler's reflection formula $\Gamma(1-z)\Gamma(z) = \frac{\pi}{\sin(\pi z)}$ and the duplication formula $\Gamma(z)\Gamma\left(z+\frac{1}{2}\right) = 2^{1-2z}\sqrt{\pi}\Gamma(2z)$ are handy.*

The *fractional power* estimator is defined in Lemma 2. See the proof in Appendix A.

**Lemma 2** *Denoted by $\hat{d}_{(\alpha),fp}$, the fractional power estimator is defined as*

$$\hat{d}_{(\alpha),fp} = \left( \frac{1}{k} \frac{\sum_{j=1}^{k} |x_j|^{\lambda^*\alpha}}{\frac{2}{\pi}\Gamma(1-\lambda^*)\Gamma(\lambda^*\alpha)\sin\left(\frac{\pi}{2}\lambda^*\alpha\right)} \right)^{1/\lambda^*} \times$$

$$\left( 1 - \frac{1}{k}\frac{1}{2\lambda^*}\left(\frac{1}{\lambda^*}-1\right)\left(\frac{\frac{2}{\pi}\Gamma(1-2\lambda^*)\Gamma(2\lambda^*\alpha)\sin(\pi\lambda^*\alpha)}{\left[\frac{2}{\pi}\Gamma(1-\lambda^*)\Gamma(\lambda^*\alpha)\sin\left(\frac{\pi}{2}\lambda^*\alpha\right)\right]^2}-1\right)\right), \qquad (12)$$

*where*

$$\lambda^* = \operatorname*{argmin}_{-\frac{1}{2\alpha}\lambda<\frac{1}{2}} g\left(\lambda;\alpha\right), \qquad g\left(\lambda;\alpha\right) = \frac{1}{\lambda^2}\left(\frac{\frac{2}{\pi}\Gamma(1-2\lambda)\Gamma(2\lambda\alpha)\sin(\pi\lambda\alpha)}{\left[\frac{2}{\pi}\Gamma(1-\lambda)\Gamma(\lambda\alpha)\sin\left(\frac{\pi}{2}\lambda\alpha\right)\right]^2}-1\right). \qquad (13)$$

*Asymptotically (i.e., as $k \to \infty$), the bias and variance of $\hat{d}_{(\alpha),fp}$ are*

$$E\left(\hat{d}_{(\alpha),fp}\right) - d_{(\alpha)} = O\left(\frac{1}{k^2}\right), \qquad (14)$$

$$Var\left(\hat{d}_{(\alpha),fp}\right) = d_{(\alpha)}^2 \frac{1}{k}\frac{1}{\lambda^{*2}}\left(\frac{\frac{2}{\pi}\Gamma(1-2\lambda^*)\Gamma(2\lambda^*\alpha)\sin(\pi\lambda^*\alpha)}{\left[\frac{2}{\pi}\Gamma(1-\lambda^*)\Gamma(\lambda^*\alpha)\sin\left(\frac{\pi}{2}\lambda^*\alpha\right)\right]^2}-1\right) + O\left(\frac{1}{k^2}\right). \qquad (15)$$

Note that in calculating $\hat{d}_{(\alpha),fp}$, the real computation only involves $\left(\sum_{j=1}^{k}|x_j|^{\lambda^*\alpha}\right)^{1/\lambda^*}$, because all other terms are basically constants and can be pre-computed.

Figure 1(a) plots $g\left(\lambda;\alpha\right)$ as a function of $\lambda$ for many different values of $\alpha$. Figure 1(b) plots the optimal $\lambda^*$ as a function of $\alpha$. We can see that $g\left(\lambda;\alpha\right)$ is a convex function of $\lambda$ and $-1 < \lambda^* < \frac{1}{2}$ (except for $\alpha = 2$), which will be proved in Lemma 3.

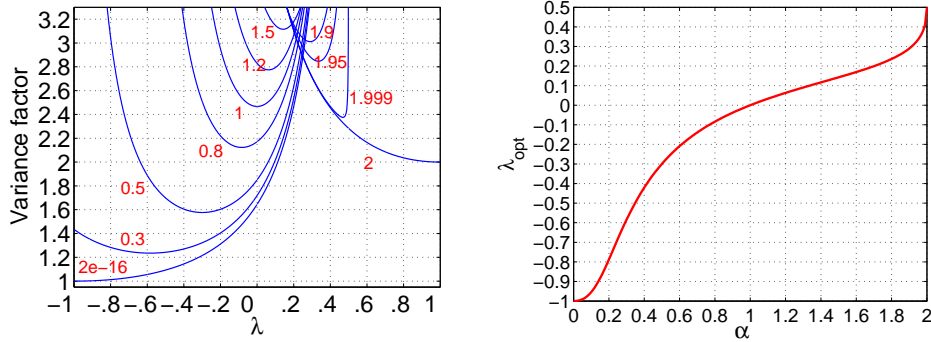

Figure 1: Left panel plots the variance factor $g\left(\lambda;\alpha\right)$ as functions of $\lambda$ for different $\alpha$, illustrating $g\left(\lambda;\alpha\right)$ is a convex function of $\lambda$ and the optimal solution (lowest points on the curves) are between -1 and 0.5 ($\alpha < 2$). Note that there is a discontinuity between $\alpha \to 2-$ and $\alpha = 2$. Right panel plots the optimal $\lambda^*$ as a function of $\alpha$. Since $\alpha = 2$ is not included, we only see $\lambda^* < 0.5$ in the figure.

## 3.1 Special cases

The discontinuity, $\lambda^*(2-) = 0.5$ and $\lambda^*(2) = 1$, reflects the fact that, for $x \sim S(\alpha,d)$, $\mathrm{E}\left(|x|^\lambda\right)$ exists for $-1 < \lambda < \alpha$ when $\alpha < 2$ and exists for any $\lambda > -1$ when $\alpha = 2$.

When $\alpha = 2$, since $\lambda^*(2) = 1$, the *fractional power* estimator becomes $\frac{1}{k}\sum_{j=1}^{k}|x_j|^2$, i.e., the *arithmetic mean* estimator. We will from now on only consider $0 < \alpha < 2$.

when $\alpha \to 0+$, since $\lambda^*(0+) = -1$, the *fractional power* estimator approaches the *harmonic mean* estimator, which is asymptotically optimal when $\alpha = 0+$ [13].

When $\alpha \to 1$, since $\lambda^*(1) = 0$ in the limit, the *fractional power* estimator has the same asymptotic variance as the *geometric mean estimator*.

## 3.2 The Asymptotic (Cramér-Rao) Efficiency

For an estimator $\hat{d}_{(\alpha)}$, its variance, under certain regularity condition, is lower-bounded by the Information inequality (also known as the Cramér-Rao bound) [11, Chapter 2], i.e., $\mathrm{Var}\left(\hat{d}_{(\alpha)}\right) \geq \frac{1}{k\mathrm{I}(\alpha)}$. The Fisher Information $\mathrm{I}(\alpha)$ can be approximated by computationally intensive procedures [19].

When $\alpha = 2$, it is well-known that the *arithmetic mean* estimator attains the Cramér-Rao bound. When $\alpha = 0+$, [13] has shown that the *harmonic mean* estimator is also asymptotically optimal. Therefore, our *fractional power* estimator achieves the Cramér-Rao bound, exactly when $\alpha = 2$, and asymptotically when $\alpha = 0+$.

The asymptotic (Cramér-Rao) efficiency is defined as the ratio of $\frac{1}{k\mathrm{I}(\alpha)}$ to the asymptotic variance of $\hat{d}_{(\alpha)}$ ($d_{(\alpha)} = 1$ for simplicity). Figure 2 plots the efficiencies for all estimators we have mentioned, illustrating that the *fractional power* estimator is near-optimal in a wide range of $\alpha$.

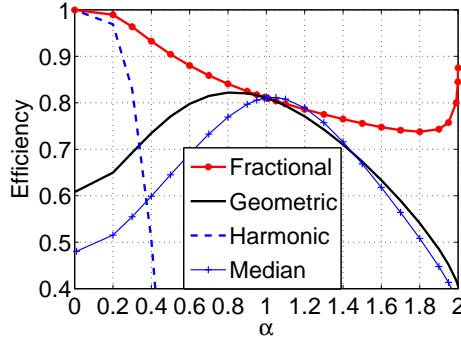

Figure 2: The asymptotic Cramér-Rao efficiencies of various estimators for $0 < \alpha < 2$, which are the ratios of $\frac{1}{k\mathrm{I}(\alpha)}$ to the asymptotic variances of the estimators. Here $k$ is the sample size and $\mathrm{I}(\alpha)$ is the Fisher Information (we use the numeric values in [19]). The asymptotic variance of the *sample median* estimator $\hat{d}_{(\alpha),me}$ is computed from known statistical theory for sample quantiles. We can see that the *fractional power* estimator $\hat{d}_{(\alpha),fp}$ is close to be optimal in a wide range of $\alpha$; and it always outperforms both the *geometric mean* and the *harmonic mean* estimators. Note that since we only consider $\alpha < 2$, the efficiency of $\hat{d}_{(\alpha),fp}$ does not achieve $100\%$ when $\alpha \to 2-$.

## 3.3 Theoretical Properties

We can show that, when computing the *fractional power* estimator $\hat{d}_{(\alpha),fp}$, to find the optimal $\lambda^*$ only involves searching for the minimum on a convex curve in the narrow range $\lambda^* \in \left(\max\left\{-1, -\frac{1}{2\alpha}\right\}, 0.5\right)$. These properties theoretically ensure that the new estimator is well-defined and is numerically easy to compute. The proof of Lemma 3 is briefly sketched in Appendix B.

**Lemma 3** *Part 1:*
$$g\left(\lambda; \alpha\right) = \frac{1}{\lambda^2}\left(\frac{\frac{2}{\pi}\Gamma(1-2\lambda)\Gamma(2\lambda\alpha)\sin\left(\pi\lambda\alpha\right)}{\left[\frac{2}{\pi}\Gamma(1-\lambda)\Gamma(\lambda\alpha)\sin\left(\frac{\pi}{2}\lambda\alpha\right)\right]^2} - 1\right), \tag{16}$$

*is a convex function of $\lambda$.*

*Part 2: For $0 < \alpha < 2$, the optimal $\lambda^* = \underset{-\frac{1}{2\alpha}\lambda<\frac{1}{2}}{\mathrm{argmin}}\, g\left(\lambda; \alpha\right)$, satisfies $-1 < \lambda^* < 0.5$.*

## 3.4 Comparing Variances at Finite Samples

It is also important to understand the small sample performance of the estimators. Figure 3 plots the empirical mean square errors (MSE) from simulations for the *fractional power* estimator, the *harmonic mean* estimator, and the *sample median* estimator. The MSE for the *geometric mean* estimators can be computed exactly without simulations.

Figure 3 indicates that the *fractional power* estimator $\hat{d}_{(\alpha),fp}$ also has good small sample performance unless $\alpha$ is close to 2. After $k \geq 50$, the advantage of $\hat{d}_{(\alpha),fp}$ becomes noticeable even when $\alpha$ is very close to 2. It is also clear that the *sample median* estimator has poor small sample performance; but even at very large $k$, its performance is not that good except when $\alpha$ is about 1.

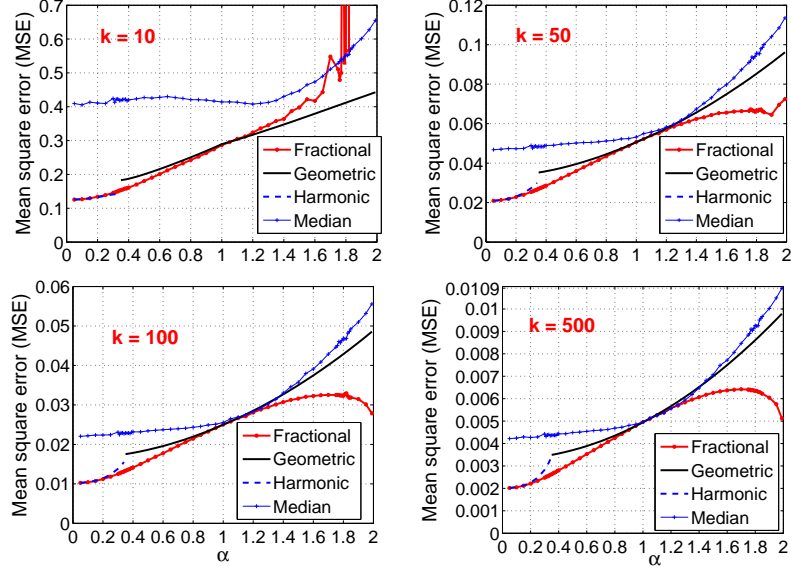

Figure 3: We simulate the mean square errors (MSE) ($10^6$ simulations at every $\alpha$ and $k$) for the *harmonic mean* estimator ($0 < \alpha \le 0.344$ only) and the *fractional power* estimator. We compute the MSE exactly for the *geometric mean* estimator (for $0.344\alpha < 2$). The *fractional power* has good accuracy (small MSE) at reasonable sample sizes (e.g., $k \ge 50$). But even at small samples (e.g., $k = 10$), it is quite accurate except when $\alpha$ approaches 2.

## 4 Discussion

The *fractional power* estimator $\hat{d}_{(\alpha),fp} \propto \left( \sum_{j=1}^{k} |x_j|^{\lambda^* \alpha} \right)^{1/\lambda^*}$ can be treated as a *linear estimator* in because the power $1/\lambda^*$ is just a constant. However, $\sum_{j=1}^{k} |x_j|^{\lambda^* \alpha}$ is not a metric because $\lambda^* \alpha < 1$, as shown in Lemma 3. Thus our result does not conflict the celebrated *impossibility result* [3], which proved that there is no hope to recover the original $l_1$ distances using *linear projections* and *linear estimators* without incurring large errors.

Although the *fractional power* estimator achieves near-optimal asymptotic variance, analyzing its tail bounds does not appear straightforward. In fact, when $\alpha$ approaches 2, this estimator does not have finite moments much higher than the second order, suggesting poor tail behavior. Our additional simulations (not included in this paper) indicate that $\hat{d}_{(\alpha),fp}$ still has comparable tail probability behavior as the *geometric mean* estimator, when $\alpha \le 1$.

Finally, we should mention that the method of *stable random projections* does not take advantage of the data sparsity while high-dimensional data (e.g., text data) are often highly sparse. A new method call *Conditional Random Sampling (CRS)* [14–16] may be more preferable in highly sparse data.

## 5 Conclusion

In massive datasets such as the Web and massive data streams, dimension reductions are often critical for many applications including clustering, classifications, recommendation systems, and Web search, because the data size may be too large for the physical memory or even for the hard disk and sometimes only one pass of the data can be afforded for building statistical learning models.

While there are already many papers on dimension reductions in the $l_2$ norm, this paper focuses on the $l_\alpha$ norm for $0 < \alpha \le 2$ using *stable random projections*, as it has become increasingly popular in machine learning to consider the $l_\alpha$ norm other than $l_2$. It is also possible to treat $\alpha$ as an additional *tuning* parameter and re-run the learning algorithms many times for better performance.

Our main contribution is the *fractional power* estimator for *stable random projections*. This estimator, with a unified format for all $0 < \alpha \le 2$, is computationally efficient and (surprisingly) is also near-optimal in terms of the asymptotic variance. We also prove some important theoretical properties (variance, convexity, etc.) to show that this estimator is well-behaved. We expect that this work will help advance the state-of-the-art of dimension reductions in the $l_\alpha$ norms.

# A    Proof of Lemma 2

By Lemma 1, we first seek an unbiased estimator of of $d_{(\alpha)}^\lambda$, denoted by $\hat{R}_{(\alpha),\lambda}$,

$$\hat{R}_{(\alpha),\lambda} = \frac{1}{k} \frac{\sum_{j=1}^k |x_j|^{\lambda\alpha}}{\frac{2}{\pi}\Gamma(1-\lambda)\Gamma(\lambda\alpha)\sin\left(\frac{\pi}{2}\lambda\alpha\right)}, \qquad -1/\alpha < \lambda < 1$$

whose variance is

$$\mathrm{Var}\left(\hat{R}_{(\alpha),\lambda}\right) = \frac{d_{(\alpha)}^{2\lambda}}{k}\left(\frac{\frac{2}{\pi}\Gamma(1-2\lambda)\Gamma(2\lambda\alpha)\sin(\pi\lambda\alpha)}{\left[\frac{2}{\pi}\Gamma(1-\lambda)\Gamma(\lambda\alpha)\sin\left(\frac{\pi}{2}\lambda\alpha\right)\right]^2}-1\right), \qquad -\frac{1}{2\alpha} < \lambda < \frac{1}{2}$$

A biased estimator of $d_{(\alpha)}$ would be simply $\left(\hat{R}_{(\alpha),\lambda}\right)^{1/\lambda}$, which has $O\left(\frac{1}{k}\right)$ bias. This bias can be removed to an extent by Taylor expansions [11, Theorem 6.1.1]. While it is well-known that bias-corrections are not always beneficial because of the bias-variance trade-off phenomenon, in our case, it is a good idea to conduct the bias-correction because the function $f(x) = x^{1/\lambda}$ is convex for $x > 0$. Note that $f'(x) = \frac{1}{\lambda}x^{1/\lambda-1}$ and $f''(x) = \frac{1}{\lambda}\left(\frac{1}{\lambda}-1\right)x^{1/\lambda-2} > 0$, assuming $-\frac{1}{2\alpha} < \lambda < \frac{1}{2}$. Because $f(x)$ is convex, removing the $O\left(\frac{1}{k}\right)$ bias will also lead to a smaller variance.

We call this new estimator the "fractional power" estimator:

$$\hat{d}_{(\alpha),fp,\lambda} = \left(\hat{R}_{(\alpha),\lambda}\right)^{1/\lambda} - \frac{\mathrm{Var}\left(\hat{R}_{(\alpha),\lambda}\right)}{2}\frac{1}{\lambda}\left(\frac{1}{\lambda}-1\right)\left(d_{(\alpha)}^\lambda\right)^{1/\lambda-2}$$

$$= \left(\frac{1}{k}\frac{\sum_{j=1}^k |x_j|^{\lambda\alpha}}{\frac{2}{\pi}\Gamma(1-\lambda)\Gamma(\lambda\alpha)\sin\left(\frac{\pi}{2}\lambda\alpha\right)}\right)^{1/\lambda}\left(1 - \frac{1}{k}\frac{1}{2\lambda}\left(\frac{1}{\lambda}-1\right)\left(\frac{\frac{2}{\pi}\Gamma(1-2\lambda)\Gamma(2\lambda\alpha)\sin(\pi\lambda\alpha)}{\left[\frac{2}{\pi}\Gamma(1-\lambda)\Gamma(\lambda\alpha)\sin\left(\frac{\pi}{2}\lambda\alpha\right)\right]^2}-1\right)\right),$$

where we plug in the estimated $d_{(\alpha)}^\lambda$. The asymptotic variance would be

$$\mathrm{Var}\left(\hat{d}_{(\alpha),fp,\lambda}\right) = \mathrm{Var}\left(\hat{R}_{(\alpha),\lambda}\right)\left(\frac{1}{\lambda}\left(d_{(\alpha)}^\lambda\right)^{1/\lambda-1}\right)^2 + O\left(\frac{1}{k^2}\right)$$

$$= d_{(\alpha)}^2 \frac{1}{\lambda^2 k}\left(\frac{\frac{2}{\pi}\Gamma(1-2\lambda)\Gamma(2\lambda\alpha)\sin(\pi\lambda\alpha)}{\left[\frac{2}{\pi}\Gamma(1-\lambda)\Gamma(\lambda\alpha)\sin\left(\frac{\pi}{2}\lambda\alpha\right)\right]^2}-1\right) + O\left(\frac{1}{k^2}\right).$$

The optimal $\lambda$, denoted by $\lambda^*$, is then

$$\lambda^* = \underset{-\frac{1}{2\alpha}\lambda<\frac{1}{2}}{\mathrm{argmin}}\left\{\frac{1}{\lambda^2}\left(\frac{\frac{2}{\pi}\Gamma(1-2\lambda)\Gamma(2\lambda\alpha)\sin(\pi\lambda\alpha)}{\left[\frac{2}{\pi}\Gamma(1-\lambda)\Gamma(\lambda\alpha)\sin\left(\frac{\pi}{2}\lambda\alpha\right)\right]^2}-1\right)\right\}.$$

# B    Proof of Lemma 3

We sketch the basic steps; and we direct readers to the additional supporting material for more detail.

We use the infinite-product representations of the Gamma and sine functions [8, 8.322,1.431.1],

$$\Gamma(z) = \frac{\exp(-\gamma_e z)}{z}\prod_{s=1}^\infty\left(1+\frac{z}{s}\right)^{-1}\exp\left(\frac{z}{s}\right), \qquad \sin(z) = z\prod_{s=1}^\infty\left(1-\frac{z^2}{s^2\pi^2}\right),$$

to re-write $g(\lambda;\alpha)$ as

$$g(\lambda;\alpha) = \frac{1}{\lambda^2}(M(\lambda;\alpha)-1) = \frac{1}{\lambda^2}\left(\prod_{s=1}^\infty f_s(\lambda;\alpha)-1\right),$$

$$f_s(\lambda;\alpha) = \left(1-\frac{\lambda}{s}\right)^2\left(1+\frac{2\lambda\alpha}{s}\right)^{-1}\left(1-\frac{\lambda\alpha}{s}\right)\left(1+\frac{\lambda\alpha}{s}\right)^3\left(1-\frac{\lambda^2\alpha^2}{4s^2}\right)^{-2}\left(1-\frac{2\lambda}{s}\right)^{-1}.$$

With respect to $\lambda$, the first two derivatives of $g(\lambda;\alpha)$ are

$$\frac{\partial g}{\partial\lambda} = \frac{1}{\lambda^2}\left(-\frac{2}{\lambda}(M-1)+\sum_{s=1}^\infty\frac{\partial\log f_s}{\partial\lambda}M\right).$$

$$\frac{\partial^2 g}{\partial\lambda^2} = \frac{M}{\lambda^2}\left(\frac{6}{\lambda^2}+\sum_{s=1}^\infty\frac{\partial^2\log f_s}{\partial\lambda^2}+\left(\sum_{s=1}^\infty\frac{\partial\log f_s}{\partial\lambda}\right)^2-\frac{4}{\lambda}\sum_{s=1}^\infty\frac{\partial\log f_s}{\partial\lambda}\right)-\frac{6}{\lambda^4}.$$

Also,

$$\sum_{s=1}^{\infty} \frac{\partial \log f_s}{\partial \lambda} = 2\lambda \sum_{s=1}^{\infty} \frac{1}{s^2 - 3s\lambda + 2\lambda^2} + \alpha^2 \left( \frac{2}{4s^2 - \lambda^2\alpha^2} + \frac{1}{s^2 + 3s\lambda\alpha + 2\lambda^2\alpha^2} - \frac{1}{s^2 - \lambda^2\alpha^2} \right),$$

$$\sum_{s=1}^{\infty} \frac{\partial^2 \log f_s}{\partial \lambda^2} = \sum_{s=1}^{\infty} \frac{-2}{(s-\lambda)^2} + \frac{4}{(s-2\lambda)^2} + \frac{2\alpha^2}{(2s-\lambda\alpha)^2} - \frac{\alpha^2}{(s-\lambda\alpha)^2} - \frac{3\alpha^2}{(s+\lambda\alpha)^2} + \frac{4\alpha^2}{(s+2\lambda\alpha)^2} + \frac{2\alpha^2}{(2s+\lambda\alpha)^2}$$

$$\sum_{s=1}^{\infty} \frac{\partial^3 \log f_s}{\partial \lambda^3}$$

$$= \sum_{s=1}^{\infty} \frac{4}{(s-\lambda)^3} + \frac{16}{(s-2\lambda)^3} + 2\alpha^3 \left( \frac{2}{(2s-\lambda\alpha)^3} - \frac{1}{(s-\lambda\alpha)^3} + \frac{3}{(s+\lambda\alpha)^3} - \frac{8}{(s+2\lambda\alpha)^3} - \frac{2}{(2s+\lambda\alpha)^3} \right),$$

$$\sum_{s=1}^{\infty} \frac{\partial^4 \log f_s}{\partial \lambda^4}$$

$$= \sum_{s=1}^{\infty} \frac{-12}{(s-\lambda)^4} + \frac{96}{(s-2\lambda)^4} + 6\alpha^4 \left( \frac{2}{(2s-\lambda\alpha)^4} - \frac{1}{(s-\lambda\alpha)^4} - \frac{3}{(s+\lambda\alpha)^4} + \frac{16}{(s+2\lambda\alpha)^4} + \frac{2}{(2s+\lambda\alpha)^4} \right).$$

To show $\frac{\partial^2 g}{\partial \lambda^2} > 0$, it suffices to show $\lambda^4 \frac{\partial^2 g}{\partial \lambda^2} > 0$, which can shown based on its own second derivative (and hence we need $\sum_{s=1}^{\infty} \frac{\partial^4 \log f_s}{\partial \lambda^4}$). Here we consider $\lambda \neq 0$ to avoid triviality. To complete the proof, we use some properties of the Riemann's Zeta function and the infinite countability.

Next, we show that $\lambda^* < -1$ does not satisfy $\left. \frac{\partial g(\lambda;\alpha)}{\partial \lambda} \right|_{\lambda^*} = 0$, which is equivalent to $h(\lambda^*) = 1$,

$$h(\lambda^*) = M(\lambda^*) \left( 1 - \frac{\lambda^*}{2} \sum_{s=1}^{\infty} \left. \frac{\partial \log f_s}{\partial \lambda} \right|_{\lambda^*} \right) = 1,$$

We show that when $\lambda < -1$, $\frac{\partial h}{\partial \lambda} > 0$, i.e., $h(\lambda) < h(-1)$. We then show $\frac{\partial h(-1)}{\partial \alpha} < 0$ for $0 < \alpha < 0.5$; and hence $h(-1;\alpha) < h(-1;0+) = 1$. Therefore, we must have $\lambda^* > -1$.

## References

[1] C. Aggarwal, editor. *Data Streams: Models and Algorithms*. Springer, New York, NY, 2007.

[2] B. Babcock, S. Babu, M. Datar, R. Motwani, and J. Widom. Models and issues in data stream systems. In *PODS*, 1–16, 2002.

[3] B. Brinkman and M. Charikar. On the impossibility of dimension reduction in $l_1$. *Journal of ACM*, 52(2):766–788, 2005.

[4] O. Chapelle, P. Haffner, and V. Vapnik. Support vector machines for histogram-based image classification. *IEEE Trans. Neural Networks*, 10(5):1055–1064, 1999.

[5] G. Cormode, M. Datar, P. Indyk, and S. Muthukrishnan. Comparing data streams using hamming norms (how to zero in). In *VLDB*, 335–345, 2002.

[6] D. Donoho. Compressed sensing. *IEEE Trans. Inform. Theory*, 52(4):1289–1306, 2006.

[7] E. Fama and R. Roll. Parameter estimates for symmetric stable distributions. *JASA*, 66(334):331–338, 1971.

[8] I. Gradshteyn and I. Ryzhik. *Table of Integrals, Series, and Products*. Academic Press, New York, fifth edition, 1994.

[9] P. Indyk. Stable distributions, pseudorandom generators, embeddings, and data stream computation. *Journal of ACM*, 53(3):307–323, 2006.

[10] W. Johnson and J. Lindenstrauss. Extensions of Lipschitz mapping into Hilbert space. *Contemporary Mathematics*, 26:189–206, 1984.

[11] E. Lehmann and G. Casella. *Theory of Point Estimation*. Springer, New York, NY, second edition, 1998.

[12] E. Leopold and J. Kindermann. Text categorization with support vector machines. how to represent texts in input space? *Machine Learning*, 46(1-3):423–444, 2002.

[13] P. Li. Estimators and tail bounds for dimension reduction in $l_\alpha$ $(0 < \alpha \leq 2)$ using stable random projections. In *SODA*, 2008.

[14] P. Li and K. Church. Using sketches to estimate associations. In *HLT/EMNLP*, 708–715, 2005.

[15] P. Li and K. Church. A sketch algorithm for estimating two-way and multi-way associations. *Computational Linguistics*, 33(3):305–354, 2007.

[16] P. Li, K. Church, and T. Hastie. Conditional random sampling: A sketch-based sampling technique for sparse data. In *NIPS*, 873–880, 2007.

[17] P. Li, T. Hastie, and K. Church. Improving random projections using marginal information. In *COLT*, 635–649, 2006.

[18] P. Li, T. Hastie, and K. Church. Nonlinear estimators and tail bounds for dimensional reduction in $l_1$ using cauchy random projections. *Journal of Machine Learning Research* (To appear) .

[19] M. Matsui and A. Takemura. Some improvements in numerical evaluation of symmetric stable density and its derivatives. *Communications on Statistics-Theory and Methods*, 35(1):149–172, 2006.

[20] J. McCulloch. Simple consistent estimators of stable distribution parameters. *Communications on Statistics-Simulation*, 15(4):1109–1136, 1986.

[21] J. Rennie, L. Shih, J. Teevan, and D. Karger. Tackling the poor assumptions of naive Bayes text classifiers. In *ICML*, 616–623, 2003.

[22] S. Vempala. *The Random Projection Method*. American Mathematical Society, Providence, RI, 2004.

[23] J. Zhu, S. Rosset, T. Hastie, and R. Tibshirani. 1-norm support vector machines. In *NIPS*, Vancouver, 2003.

[24] V. M. Zolotarev. *One-dimensional Stable Distributions*. American Mathematical Society, Providence, RI, 1986.
